# Nonparametric Bayesian Policy Priors for Reinforcement Learning

**Finale Doshi-Velez, David Wingate, Nicholas Roy and Joshua Tenenbaum**
Massachusetts Institute of Technology
Cambridge, MA 02139
{finale,wingated,nickroy,jbt}@csail.mit.edu

## Abstract

We consider reinforcement learning in partially observable domains where the agent can query an expert for demonstrations. Our nonparametric Bayesian approach combines model knowledge, inferred from expert information and independent exploration, with policy knowledge inferred from expert trajectories. We introduce priors that bias the agent towards models with both simple representations and simple policies, resulting in improved policy and model learning.

## 1 Introduction

We address the reinforcement learning (RL) problem of finding a good policy in an unknown, stochastic, and partially observable domain, given both data from independent exploration and expert demonstrations. The first type of data, from independent exploration, is typically used by model-based RL algorithms [1, 2, 3, 4] to learn the world's dynamics. These approaches build models to predict observation and reward data given an agent's actions; the action choices themselves, since they are made by the agent, convey no statistical information about the world. In contrast, imitation and inverse reinforcement learning [5, 6] use expert trajectories to learn reward models. These approaches typically assume that the world's dynamics is known.

We consider cases where we have data from both independent exploration and expert trajectories. Data from independent observation gives direct information about the dynamics, while expert demonstrations show outputs of good policies and thus provide indirect information about the underlying model. Similarly, rewards observed during independent exploration provide indirect information about good policies. Because dynamics and policies are linked through a complex, nonlinear function, leveraging information about both these aspects at once is challenging. However, we show that using both data improves model-building and control performance.

We use a Bayesian model-based RL approach to take advantage of both forms of data, applying Bayes rule to write a posterior over models $M$ given data $D$ as $p(M|D) \propto p(D|M)p(M)$. In previous work [7, 8, 9, 10], the model prior $p(M)$ was defined as a distribution directly on the dynamics and rewards models, making it difficult to incorporate expert trajectories. Our main contribution is a new approach to defining this prior: our prior uses the assumption that the expert knew something about the world model when computing his optimal policy. Different forms of these priors lead us to three different learning algorithms: (1) if we know the expert's planning algorithm, we can sample models from $p(M|D)$, invoke the planner, and weigh models given how likely it is the planner's policy generated the expert's data; (2) if, instead of a planning algorithm, we have a *policy prior*, we can similarly weight world models according to how likely it is that probable policies produced the expert's data; and (3) we can search directly in the policy space guided by probable models.

We focus on reinforcement learning in discrete action and observation spaces. In this domain, one of our key technical contributions is the insight that the Bayesian approach used for building models of transition dynamics can also be used as policy priors, if we exchange the typical role of actions and

observations. For example, algorithms for learning partially observable Markov decision processes (POMDPs) build models that output observations and take in actions as exogenous variables. If we reverse their roles, the observations become the exogenous variables, and the model-learning algorithm is exactly equivalent to learning a finite-state controller [11]. By using nonparametric priors [12], our agent can scale the sophistication of its policies and world models based on the data.

Our framework has several appealing properties. First, our choices for the policy prior and a world model prior can be viewed as a joint prior which introduces a bias for world models which are both simple and easy to control. This bias is especially beneficial in the case of direct policy search, where it is easier to search directly for good controllers than it is to first construct a complete POMDP model and then plan with it. Our method can also be used with approximately optimal expert data; in these cases the expert data can be used to bias which models are likely but not set hard constraints on the model. For example, in Sec. 4 an application where we extract the essence of a good controller from good—but not optimal—trajectories generated by a randomized planning algorithm.

## 2   Background

A partially observable Markov decision process (POMDP) model $M$ is an n-tuple $\{S,A,O,T,\Omega,R,\gamma\}$. $S$, $A$, and $O$ are sets of states, actions, and observations. The state transition function $T(s'|s,a)$ defines the distribution over next-states $s'$ to which the agent may transition after taking action $a$ from state $s$. The observation function $\Omega(o|s',a)$ is a distribution over observations $o$ that may occur in state $s'$ after taking action $a$. The reward function $R(s,a)$ specifies the immediate reward for each state-action pair, while $\gamma \in [0,1)$ is the discount factor. We focus on learning discrete state, observation, and action spaces.

**Bayesian RL**   In Bayesian RL, the agent starts with a prior distribution $P(M)$ over possible POMDP models. Given data $D$ from an unknown , the agent can compute a posterior over possible worlds $P(M|D) \propto P(D|M)P(M)$. The model prior can encode both vague notions, such as "favor simpler models," and strong structural assumptions, such as topological constraints among states. Bayesian nonparametric approaches are well-suited for partially observable environments because they can also infer the dimensionality of the underlying state space. For example, the recent infinite POMDP (iPOMDP) [12] model, built from HDP-HMMs [13, 14], places prior over POMDPs with infinite states but introduces a strong locality bias towards exploring only a few.

The decision-theoretic approach to acting in the Bayesian RL setting is to treat the model $M$ as additional hidden state in a larger "model-uncertainty" POMDP and plan in the joint space of models and states. Here, $P(M)$ represents a belief over models. Computing a Bayes-optimal policy is computationally intractable; methods approximate the optimal policy by sampling a single model and following that model's optimal policy for a fixed period of time [8]; by sampling multiple models and choosing actions based on a vote or stochastic forward search [1, 4, 12, 2]; and by trying to approximate the value function for the full model-uncertainty POMDP analytically [7]. Other approaches [15, 16, 9] try to balance the off-line computation of a good policy (the computational complexity) and the cost of getting data online (the sample complexity).

**Finite State Controllers**   Another possibility for choosing actions—including in our partially-observable reinforcement learning setting—is to consider a parametric family of policies, and attempt to estimate the optimal policy parameters from data. This is the approach underlying, for example, much work on policy gradients. In this work, we focus on the popular case of a finite-state controller, or FSC [11]. An FSC consists of the n-tuple $\{N,A,O,\pi,\beta\}$. $N$, $A$, and $O$ are sets of nodes, actions, and observations. The node transition function $\beta(n'|n,o)$ defines the distribution over next-nodes $n'$ to which the agent may transition after taking action $a$ from node $n$. The policy function $\pi(a|n)$ is a distribution over actions that the finite state controller may output in node $n$. Nodes are discrete; we again focus on discrete observation and action spaces.

## 3   Nonparametric Bayesian Policy Priors

We now describe our framework for combining world models and expert data. Recall that our key assumption is that the expert used knowledge about the underlying world to derive his policy. Fig. 1

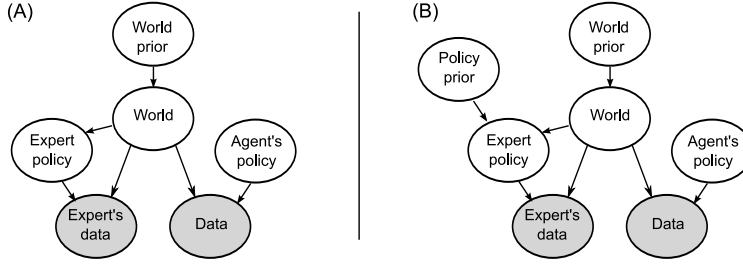

Figure 1: Two graphical models of expert data generation. Left: the prior only addresses world dynamics and rewards. Right: the prior addresses both world dynamics and controllable policies.

shows the two graphical models that summarize our approaches. Let $M$ denote the (unknown) world model. Combined with the world model $M$, the expert's policy $\pi_e$ and agent's policy $\pi_a$ produce the expert's and agent's data $D_e$ and $D_a$. The data consist of a sequence of *histories*, where a history $h_t$ is a sequence of actions $a_1, \cdots, a_t$, observations $o_1, \cdots, o_t$, and rewards $r_1, \cdots, r_t$. The agent has access to all histories, but the true world model and optimal policy are hidden.

Both graphical models assume that a particular world $M$ is sampled from a prior over POMDPs, $g_M(M)$. In what would be the standard application of Bayesian RL with expert data (Fig. 1(a)), the prior $g_M(M)$ fully encapsulates our initial belief over world models. An expert, who knows the true world model $M$, executes a planning algorithm $\texttt{plan}(M)$ to construct an optimal policy $\pi_e$. The expert then executes the policy to generate expert data $D_e$, distributed according to $p(D_e|M, \pi_e)$, where $\pi_e = \texttt{plan}(M)$.

However, the graphical model in Fig. 1(a) does not easily allow us to encode a prior bias toward more controllable world models. In Fig. 1(b), we introduce a new graphical model in which we allow additional parameters in the distribution $p(\pi_e)$. In particular, if we choose a distribution of the form

$$p(\pi_e|M) \propto f_M(\pi_e)g_\pi(\pi_e) \tag{1}$$

where we interpret $g_\pi(\pi_e)$ as a *prior over policies* and $f_M(\pi_e)$ as a *likelihood of a policy given a model*. We can write the distribution over world models as

$$p(M) \propto \int_{\pi_e} f_M(\pi_e)g_\pi(\pi_e)g_M(M) \tag{2}$$

If $f_M(\pi_e)$ is a delta function on $\texttt{plan}(M)$, then the integral in Eq. 2 reduces to

$$p(M) \propto g_\pi(\pi_e^M)g_M(M) \tag{3}$$

where $\pi_e^M = \texttt{plan}(M)$, and we see that we have a prior that provides input on both the world's dynamics and the world's controllability. For example, if the policy class is the set of finite state controllers as discussed in Sec. 2, the policy prior $g_\pi(\pi_e)$ might encode preferences for a smaller number of nodes used the policy, while $g_M(M)$ might encode preferences for a smaller number of visited states in the world. The function $f_M(\pi_e)$ can also be made more general to encode how likely it is that the expert uses the policy $\pi_e$ given world model $M$.

Finally, we note that $p(D_e|M, \pi)$ factors as $p(D_e^a|\pi)p(D_e^{o,r}|M)$, where $D_e^a$ are the actions in the histories $D_e$ and $D_e^{o,r}$ are the observations and rewards. Therefore, the conditional distribution over world models given data $D_e$ and $D_a$ is:

$$p(M|D_e, D_a) \propto p(D_e^{o,r}, D_a|M)g_M(M)\int_{\pi_e} p(D_e^a|\pi_e)g_\pi(\pi_e)f_M(\pi_e) \tag{4}$$

The model in Fig. 1(a) corresponds to setting a uniform prior on $g_\pi(\pi_e)$. Similarly, the conditional distribution over policies given data $D_e$ and $D_a$ is

$$p(\pi_e|D_e, D_a) \propto g_\pi(\pi_e)p(D_e^a|\pi_e)\int_M f_M(\pi_e)p(D_e^{o,r}, D_a|M)g_M(M) \tag{5}$$

We next describe three inference approaches for using Eqs. 4 and 5 to learn.

**#1: Uniform Policy Priors (Bayesian RL with Expert Data).** If $f_M(\pi_{\mathrm{e}}) = \delta(\texttt{plan}(M))$ and we believe that all policies are equally likely (graphical model 1(a)), then we can leverage the expert's data by simply considering how well that world model's policy $\texttt{plan}(M)$ matches the expert's actions for a particular world model $M$. Eq. 4 allows us to compute a posterior over world models that accounts for the quality of this match. We can then use that posterior as part of a planner by using it to evaluate candidate actions. The expected value of an action[1] $q(a)$ with respect to this posterior is given by:

$$
\begin{aligned}
\mathbb{E}\left[q(a)\right] &= \int_M q(a|M)p(M|D_{\mathrm{e}}^{o,r}, D_{\mathrm{a}}) \\
&= \int_M q(a|M)p(D_{\mathrm{e}}^{o,r}, D_{\mathrm{a}}|M)g_M(M)p(D_{\mathrm{e}}^a|\texttt{plan}(M)) \quad (6)
\end{aligned}
$$

We assume that we can draw samples from $p(M|D_{\mathrm{e}}^{o,r}, D_{\mathrm{a}}) \propto p(D_{\mathrm{e}}^{o,r}, D_{\mathrm{a}}|M)g_M(M)$, a common assumption in Bayesian RL [12, 9]; for our iPOMDP-based case, we can draw these samples using the beam sampler of [17]. We then weight those samples by $p(D_{\mathrm{e}}^a|\pi_{\mathrm{e}})$, where $\pi_{\mathrm{e}} = \texttt{plan}(M)$, to yield the importance-weighted estimator

$$
\mathbb{E}\left[q(a)\right] \approx \sum_i q(a|M_i)p(D_{\mathrm{e}}^a|M_i, \pi_{\mathrm{e}}), \quad M_i \sim p(M|D_{\mathrm{e}}^{o,r}, D_{\mathrm{a}}).
$$

Finally, we can also sample values for $q(a)$ by first sampling a world model given the importance-weighted distribution above and recording the $q(a)$ value associated with that model.

**#2: Policy Priors with Model-based Inference.** The uniform policy prior implied by standard Bayesian RL does not allow us to encode prior biases about the policy. With a more general prior (graphical model 1(b) in Fig. 1), the expectation in Eq. 6 becomes

$$
\mathbb{E}\left[q(a)\right] = \int_M q(a|M)p(D_{\mathrm{e}}^{o,r}, D_{\mathrm{a}}|M)g_M(M)g_\pi(\texttt{plan}(M))p(D_{\mathrm{e}}^a|\texttt{plan}(M)) \quad (7)
$$

where we still assume that the expert uses an optimal policy, that is, $f_M(\pi_{\mathrm{e}}) = \delta(\texttt{plan}(M))$. Using Eq. 7 can result in somewhat brittle and computationally intensive inference, however, as we must compute $\pi_{\mathrm{e}}$ for each sampled world model $M$. It also assumes that the expert used the optimal policy, whereas a more realistic assumption might be that the expert uses a near-optimal policy.

We now discuss an alternative that relaxes $f_M(\pi_{\mathrm{e}}) = \delta(\texttt{plan}(M))$: let $f_M(\pi_{\mathrm{e}})$ be a function that prefers policies that achieve higher rewards in world model $M$: $f_M(\pi_{\mathrm{e}}) \propto \exp\left\{V(\pi_{\mathrm{e}}|M)\right\}$, where $V(\pi_{\mathrm{e}}|M)$ is the value of the policy $\pi_{\mathrm{e}}$ on world $M$; indicating a belief that the expert tends to sample policies that yield high value. Substituting this $f_M(\pi_{\mathrm{e}})$ into Eq. 4, the expected value of an action is

$$
\mathbb{E}\left[q(a)\right] = \int_{M,\pi_{\mathrm{e}}} q(a|M)p(D_{\mathrm{e}}^a|\pi_{\mathrm{e}})\exp\left\{V(\pi_{\mathrm{e}}|M)\right\}g_\pi(\pi_{\mathrm{e}})p(D_{\mathrm{e}}^{o,r}, Da|M)g_M(M)
$$

We again assume that we can draw samples from $p(M|D_{\mathrm{e}}^{o,r}, D_{\mathrm{a}}) \propto p(D_{\mathrm{e}}^{o,r}, D_{\mathrm{a}}|M)g_M(M)$, and additionally assume that we can draw samples from $p(\pi_{\mathrm{e}}|D_{\mathrm{e}}^a) \propto p(D_{\mathrm{e}}^a|\pi_{\mathrm{e}})g_\pi(\pi_{\mathrm{e}})$, yielding:

$$
\mathbb{E}\left[q(a)\right] \approx \sum_i q(a|M_i)\sum_j \exp\left\{V(\pi_{\mathrm{e}j}|M_i)\right\}, \quad M_i \sim p(M|D_{\mathrm{e}}^{o,r}, D_{\mathrm{a}}), \pi_{\mathrm{e}j} \sim p(\pi_{\mathrm{e}}|D_{\mathrm{e}}^a) \quad (8)
$$

As in the case with standard Bayesian RL, we can also use our weighted world models to draw samples from $q(a)$.

**#3: Policy Priors with Joint Model-Policy Inference.** While the model-based inference for policy priors is correct, using importance weights often suffers when the proposal distribution is not near the true posterior. In particular, sampling world models and policies—both very high dimensional objects—from distributions that ignore large parts of the evidence means that large numbers of samples may be needed to get accurate estimates. We now describe an inference approach that alternates sampling models and policies that both avoids importance sampling and can be used even

in cases where $f_M(\pi_e) = \delta(\texttt{plan}(M))$. Once we have a set of sampled models we can compute the expectation $\mathbb{E}[q(a)]$ simply as the average over the action values $q(a|M_i)$ for each sampled model.

The inference proceeds in two alternating stages: first, we sample a new policy given a sampled model. Given a world model, Eq. 5 becomes

$$p(\pi_e|D_e, D_a, M) \propto g_\pi(\pi_e)p(D_e^a|\pi_e)f_M(\pi_e) \qquad (9)$$

where making $g_\pi(\pi_e)$ and $p(D_e^a|\pi_e)$ conjugate is generally an easy design choice—for example, in Sec. 3.1, we use the iPOMDP [12] as a conjugate prior over policies encoded as finite state controllers. We then approximate $f_M(\pi_e)$ with a function in the same conjugate family: in the case of the iPOMDP prior and count data $D_e^a$, we also approximate $f_M$ with a set of Dirichlet counts scaled by some temperature parameter $a$. As $a$ is increased, we recover the desired $f_M(\pi_e) = \delta(\texttt{plan}(M))$; the initial approximation speeds up the inference and does not affect its correctness.

Next we sample a new world model given the policy. Given a policy, Eq. 4 reduces to

$$p(M|D_e, D_a) \propto p(D_e^{o,r}, D_a|M)g_M(M)f_M(\pi_e). \qquad (10)$$

We apply a Metropolis-Hastings (MH) step to sample new world models, drawing a new model $M'$ from $p(D_e^{o,r}, D_a|M)g_M(M)$ and accepting it with ratio $\frac{f_{M'}(\pi_e)}{f_M(\pi_e)}$. If $f_M(\pi_e)$ is highly peaked, then this ratio is likely to be ill-defined; as when sampling policies, we apply a tempering scheme in the inference to smooth $f_M(\pi_e)$. For example, if we desired $f_M(\pi_e) = \delta(\texttt{plan}(M))$, then we could use smoothed version $\hat{f_M}(\pi_e) \propto \exp(a \cdot (V(\pi_e|M) - V(\pi_e^M|M))^2)$, where $b$ is a temperature parameter for the inference. While applying MH can suffer from the same issues as the importance sampling in the model-based approach, Gibbs sampling new policies removes one set of proposal distributions from the inference, resulting in better estimates with fewer samples.

## 3.1 Priors over State Controller Policies

We now turn to the definition of the policy prior $p(\pi_e)$. In theory, any policy prior can be used, but there are some practical considerations. Mathematically, the policy prior serves as a regularizer to avoid overfitting the expert data, so it should encode a preference toward simple policies. It should also allow computationally tractable sampling from the posterior $p(\pi_e|D_e) \propto p(D_e|\pi_e)p(\pi_e)$.

In discrete domains, one choice for the policy prior (as well as the model prior) is the iPOMDP [12]. To use the iPOMDP as a model prior (its intended use), we treat actions as inputs and observations as outputs. The iPOMDP posits that there are an infinite number of states $s$ but a few popular states are visited most of the time; the beam sampler [17] can efficiently draw samples of state transition, observation, and reward models for visited states. Joint inference over the model parameters $T, \Omega, R$ and the state sequence $\underline{s}$ allows us to infer the number of visited states from the data.

To use the iPOMDP as a policy prior, we simply reverse the roles of actions and observations, treating the observations as inputs and the actions as outputs. Now, the iPOMDP posits that there is a state controller with an infinite number of nodes $n$, but probable polices use only a small subset of the nodes a majority of the time. We perform joint inference over the node transition and policy parameters $\beta$ and $\pi$ as well as the visited nodes $\underline{n}$. The 'policy state' representation learned is not the world state, rather it is a summary of previous observations which is sufficient to predict actions. Assuming that the training action sequences are drawn from the optimal policy, the learner will learn just enough "policy state" to control the system optimally. As in the model prior application, using the iPOMDP as a policy prior biases the agent towards simpler policies—those that visit fewer nodes—but allows the number of nodes to grow as with new expert experience.

## 3.2 Consistency and Correctness

In all three inference approaches, the sampled models and policies are an unbiased representation of the true posterior and are consistent in that in the limit of infinite samples, we will recover the true model and policy posteriors conditioned on their respective data $D_a, D_e^{o,r}$ and $D_e^a$. There are some mild conditions on the world and policy priors to ensure consistency: since the policy prior and model prior are specified independently, we require that there exist models for which both the policy prior and model prior are non-zero in the limit of data. Formally, we also require that the expert provide optimal trajectories; in practice, we see that this assumption can be relaxed.

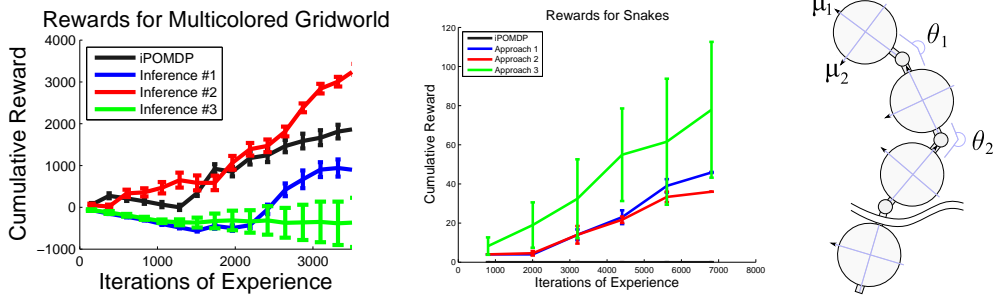

Figure 2: Learning curves for the multicolored gridworld (left) and snake (right). Error bars are 95% confidence intervals of the mean. On the far right is the snake robot.

### 3.3 Planning with Distributions over Policies and Models

All the approaches in Sec. 3 output samples of models or policies to be used for planning. As noted in Section 2, computing the Bayes optimal action is typically intractable. Following similar work [4, 1, 2, 12], we interpret these samples as beliefs. In the model-based approaches, we first solve each model (all of which are generally small) using standard POMDP planners. During the testing phase, the internal belief state of the models (in the model-based approaches) or the internal node state of the policies (in the policy-based approaches), is updated after each action-observation pair. Models are also reweighted using standard importance weights so that they continue to be an unbiased approximation of the true belief. Actions are chosen by first selecting, depending on the approach, a model or policy based on their weights, and then performing its most preferred action. While this approach is clearly approximate (it considers state uncertainty but not model uncertainty), we found empirically that this simple, fast approach to action selection produced nearly identical results to the much slower (but asymptotically Bayes optimal) stochastic forward search in [12].[2]

## 4 Experiments

We first describe a pair of demonstrations that show two important properties of using policy priors: (1) that policy priors can be useful even in the absence of expert data and (2) that our approach works even when the expert trajectories are not optimal. We then compare policy priors with the basic iPOMDP [12] and finite-state model learner trained with EM on several standard problems. In all cases, the tasks were episodic. Since episodes could be of variable length—specifically, experts generally completed the task in fewer iterations—we allowed each approach $N = 2500$ iterations, or interactions with the world, during each learning trial. The agent was provided with an expert trajectory with probability $.5\frac{n}{N}$, where $n$ was the current amount of experience. No expert trajectories were provided in the last quarter of the iterations. We ran each approach for 10 learning trials.

Models and policies were updated every 100 iterations, and each episode was capped at 50 iterations (though it could be shorter, if the task was achieved in fewer iterations). Following each update, we ran 50 test episodes (not included in the agent's experience) with the new models and policies to empirically evaluate the current value of the agents' policy. For all of the nonparametric approaches, 50 samples were collected, 10 iterations apart, after a burn-in of 500 iterations. Sampled models were solved using 25 backups of PBVI [18] with 500 sampled beliefs. One iteration of bounded policy iteration [19] was performed per sampled model. The finite-state learner was trained using $\min(25, |S|)$, where $|S|$ was the true number of underlying states. Both the nonparametric and finite learners were trained from scratch during each update; we found empirically that starting from random points made the learner more robust than starting it at potentially poor local optima.

**Policy Priors with No Expert Data**    The combined policy and model prior can be used to encode a prior bias towards models with simpler control policies. This interpretation of policy priors can

be useful even without expert data: the left pane of Fig. 2 shows the performance of the policy prior-biased approaches and the standard iPOMDP on a gridworld problem in which observations correspond to both the adjacent walls (relevant for planning) and the color of the square (not relevant for planning). This domain has 26 states, 4 colors, standard NSEW actions, and an 80% chance of a successful action. The optimal policy for this gridworld was simple: go east until the agent hits a wall, then go south. However, the varied observations made the iPOMDP infer many underlying states, none of which it could train well, and these models also confused the policy-inference in Approach 3. Without expert data, Approach 1 cannot do better than iPOMDP. By biasing the agent towards worlds that admit simpler policies, the model-based inference with policy priors (Approach 2) creates a faster learner.

**Policy Priors with Imperfect Experts**    While we focused on optimal expert data, in practice policy priors can be applied even if the expert is imperfect. Fig. 2(b) shows learning curves for a simulated snake manipulation problem with a 40-dimensional continuous state space, corresponding to (x,y) positions and velocities of 10 body segments. Actions are 9-dimensional continuous vectors, corresponding to desired joint angles between segments. The snake is rewarded based on the distance it travels along a twisty linear "maze," encouraging it to wiggle forward and turn corners.

We generated expert data by first deriving 16 motor primitives for the action space using a clustering technique on a near-optimal trajectory produced by a rapidly-exploring random tree (RRT). A reasonable—but not optimal—controller was then designed using alternative policy-learning techniques on the action space of motor primitives. Trajectories from this controller were treated as expert data for our policy prior model. Although the trajectories and primitives are suboptimal, Fig. 2(b) shows that knowledge of feasible solutions boosts performance when using the policy-based technique.

**Tests on Standard Problems**    We also tested the approaches on ten problems: tiger [20] (2 states), network [20] (7 states), shuttle [21] (8 states), an adapted version of gridworld [20] (26 states), an adapted version of follow [2] (26 states) hallway [20] (57 states), beach (100 states), rocksample(4,4) [22] (257 states), tag [18] (870 states), and image-search (16321 states). In the beach problem, the agent needed to track a beach ball on a 2D grid. The image-search problem involved identifying a unique pixel in an 8x8 grid with three type of filters with varying cost and scales. We compared our inference approaches with two approaches that did not leverage the expert data: expectation-maximization (EM) used to learn a finite world model of the correct size and the infinite POMDP [12], which placed the same nonparametric prior over world models as we did.

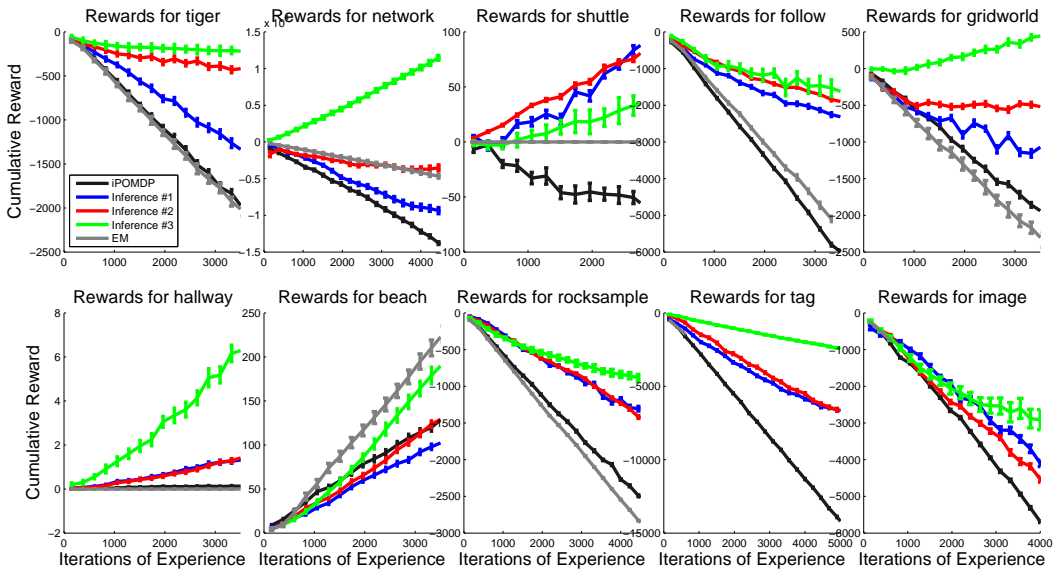

Figure 3: Performance on several standard problems, with 95% confidence intervals of the mean.

Fig. 3 shows the learning curves for our policy priors approaches (problems ordered by state space size); the cumulative rewards and final values are shown in Table 1. As expected, approaches that leverage expert trajectories generally perform better than those that ignore the near-optimality of the expert data. The policy-based approach is successful even among the larger problems. Here, even though the inferred state spaces could grow large, policies remained relatively simple. The optimization used in the policy-based approach—recall we use the stochastic search to find a probable policy—was also key to producing reasonable policies with limited computation.

| | Cumulative Reward | | | | | Final Reward | | | | |
|---|---|---|---|---|---|---|---|---|---|---|
| | iPOMDP | App. 1 | App. 2 | App. 3 | EM | iPOMDP | App. 1 | App. 2 | App. 3 | EM |
| tiger | -2.2e3 | -1.4e3 | -5.3e2 | **-2.2e2** | -3.0e3 | -2.0e1 | -1.0e1 | -2.3 | **1.6** | -2.0e1 |
| network | -1.5e4 | -6.3e3 | -2.1e3 | **1.9e4** | -2.6e3 | -1.1e1 | -1.2e1 | -4.0e-1 | **1.1e1** | -4.7 |
| shuttle | -5.3e1 | 7.9e1 | **1.5e2** | 5.1e1 | 0.0 | 1.7e-1 | 3.3e-1 | 6.5e-1 | **8.6e-1** | 0.0 |
| follow | -6.3e3 | -2.3e3 | -1.9e3 | **-1.6e3** | -5.0e3 | -5.9 | -3.1 | -1.4 | **-1.1** | -5.0 |
| gridworld | -2.0e3 | -6.2e2 | -7.0e2 | **4.6e2** | -3.7e3 | -1.3 | 5.3e-1 | 1.8 | **2.3** | -2.1 |
| hallway | 2.0e-1 | 1.4 | 1.6 | **6.6** | 0.0 | 8.6e-4 | 7.4e-3 | 1.4e-2 | **1.9e-2** | 0.0 |
| beach | 1.9e2 | 1.4e2 | 1.8e2 | 1.9e2 | **3.5e2** | 2.0e-1 | 1.1e-1 | 1.4e-1 | 2.7e-1 | **3.4e-1** |
| rocksample | -3.2e3 | -1.7e3 | -1.8e3 | **-1.0e3** | -3.5e3 | -1.6 | -5.3e-1 | -1.3 | **1.2** | -2.0 |
| tag | -1.6e4 | -6.9e3 | -7.4e3 | **-3.5e3** | - | -9.4 | -2.8 | -4.1 | **-1.7** | -9.1 |
| image | -7.8e3 | -5.3e3 | -6.1e3 | **-3.9e3** | - | -5.0 | -3.6 | -4.2 | **1.3e1** | -5.0 |

Table 1: Cumulative and final rewards on several problems. Bold values highlight best performers.

## 5    Discussion and Related Work

Several Bayesian approaches have been developed for RL in partially observable domains. These include [7], which uses a set of Gaussian approximations to allow for analytic value function updates in the POMDP space; [2], which jointly reasons over the space of Dirichlet parameters and states when planning in discrete POMDPs, and [12], which samples models from a nonparametric prior.

Both [1, 4] describe how expert data augment learning. The first [1] lets the agent to query a state oracle during the learning process. The computational benefit of a state oracle is that the information can be used to directly update a prior over models. However, in large or complex domains, the agent's state might be difficult to define. In contrast, [4] lets the agent query an expert for optimal actions. While policy information may be much easier to specify—incorporating the result of a single query into the prior over models is challenging; the particle-filtering approach of [4] can be brittle as model-spaces grow large. Our policy priors approach uses entire trajectories; by learning policies rather than single actions, we can generalize better and evaluate models more holistically. By working with models and policies, rather than just models as in [4], we can also consider larger problems which still have simple policies. Targeted criteria for asking for expert trajectories, especially one with performance guarantees such as [4], would be an interesting extension to our approach.

## 6    Conclusion

We addressed a key gap in the learning-by-demonstration literature: learning from both expert and agent data in a partially observable setting. Prior work used expert data in MDP and imitation-learning cases, but less work exists for the general POMDP case. Our Bayesian approach combined priors over the world models and policies, connecting information about world dynamics and expert trajectories. Taken together, these priors are a new way to think about specifying priors over models: instead of simply putting a prior over the dynamics, our prior provides a bias towards models with simple dynamics and simple optimal policies. We show with our approach expert data never reduces performance, and our extra bias towards controllability improves performance even without expert data. Our policy priors over nonparametric finite state controllers were relatively simple; classes of priors to address more problems is an interesting direction for future work.

## Footnotes

[1]We omit the belief over world states $b(s)$ from the equations that follow for clarity; all references to $q(a|M)$ are $q(a|b_M(s), M)$.

[2]We suspect that the reason the two planning approaches yield similar results is that the stochastic forward search never goes deep enough to discover the value of learning the model and thus acts equivalently to our sampling-based approach, which only considers the value of learning more about the underlying state.

# References

[1] R. Jaulmes, J. Pineau, and D. Precup. Learning in non-stationary partially observable Markov decision processes. ECML Workshop, 2005.

[2] Stephane Ross, Brahim Chaib-draa, and Joelle Pineau. Bayes-adaptive POMDPs. In *Neural Information Processing Systems (NIPS)*, 2008.

[3] Stephane Ross, Brahim Chaib-draa, and Joelle Pineau. Bayesian reinforcement learning in continuous POMDPs with application to robot navigation. In *ICRA*, 2008.

[4] Finale Doshi, Joelle Pineau, and Nicholas Roy. Reinforcement learning with limited reinforcement: Using Bayes risk for active learning in POMDPs. In *International Conference on Machine Learning*, volume 25, 2008.

[5] Pieter Abbeel, Morgan Quigley, and Andrew Y. Ng. Using inaccurate models in reinforcement learning. In *In International Conference on Machine Learning (ICML) Pittsburgh*, pages 1–8. ACM Press, 2006.

[6] Nathan Ratliff, Brian Ziebart, Kevin Peterson, J. Andrew Bagnell, Martial Hebert, Anind K. Dey, and Siddhartha Srinivasa. Inverse optimal heuristic control for imitation learning. In *Proc. AISTATS*, pages 424–431, 2009.

[7] P. Poupart and N. Vlassis. Model-based Bayesian reinforcement learning in partially observable domains. In *ISAIM*, 2008.

[8] M. Strens. A Bayesian framework for reinforcement learning. In *ICML*, 2000.

[9] John Asmuth, Lihong Li, Michael Littman, Ali Nouri, and David Wingate. A Bayesian sampling approach to exploration in reinforcement learning. In *Uncertainty in Artificial Intelligence (UAI)*, 2009.

[10] R. Dearden, N. Friedman, and D. Andre. Model based Bayesian exploration. pages 150–159, 1999.

[11] E. J. Sondik. *The Optimial Control of Partially Observable Markov Processes*. PhD thesis, Stanford University, 1971.

[12] Finale Doshi-Velez. The infinite partially observable Markov decision process. In Y. Bengio, D. Schuurmans, J. Lafferty, C. K. I. Williams, and A. Culotta, editors, *Advances in Neural Information Processing Systems 22*, pages 477–485. 2009.

[13] Matthew J. Beal, Zoubin Ghahramani, and Carl E. Rasmussen. The infinite hidden Markov model. In *Machine Learning*, pages 29–245. MIT Press, 2002.

[14] Yee Whye Teh, Michael I. Jordan, Matthew J. Beal, and David M. Blei. Hierarchical Dirichlet processes. *Journal of the American Statistical Association*, 101:1566–1581, 2006.

[15] Tao Wang, Daniel Lizotte, Michael Bowling, and Dale Schuurmans. Bayesian sparse sampling for on-line reward optimization. In *International Conference on Machine Learning (ICML)*, 2005.

[16] J. Zico Kolter and Andrew Ng. Near-Bayesian exploration in polynomial time. In *International Conference on Machine Learning (ICML)*, 2009.

[17] J. van Gael, Y. Saatci, Y. W. Teh, and Z. Ghahramani. Beam sampling for the infinite hidden Markov model. In *ICML*, volume 25, 2008.

[18] J. Pineau, G. Gordon, and S. Thrun. Point-based value iteration: An anytime algorithm for POMDPs. *IJCAI*, 2003.

[19] Pascal Poupart and Craig Boutilier. Bounded finite state controllers. In *Neural Information Processing Systems*, 2003.

[20] M. L. Littman, A. R. Cassandra, and L. P. Kaelbling. Learning policies for partially observable environments: scaling up. *ICML*, 1995.

[21] Lonnie Chrisman. Reinforcement learning with perceptual aliasing: The perceptual distinctions approach. In *In Proceedings of the Tenth National Conference on Artificial Intelligence*, pages 183–188. AAAI Press, 1992.

[22] T. Smith and R. Simmons. Heuristic search value iteration for POMDPs. In *Proc. of UAI 2004*, Banff, Alberta, 2004.

